# Coupled Dynamics of Fast Neurons and Slow Interactions

**A.C.C. Coolen**     **R.W. Penney**     **D. Sherrington**
Dept. of Physics - Theoretical Physics
University of Oxford
1 Keble Road, Oxford OX1 3NP, U.K.

## Abstract

A simple model of coupled dynamics of fast neurons and slow interactions, modelling self-organization in recurrent neural networks, leads naturally to an effective statistical mechanics characterized by a partition function which is an average over a replicated system. This is reminiscent of the replica trick used to study spin-glasses, but with the difference that the number of replicas has a physical meaning as the ratio of two temperatures and can be varied throughout the whole range of real values. The model has interesting phase consequences as a function of varying this ratio and external stimuli, and can be extended to a range of other models.

## 1   A SIMPLE MODEL WITH FAST DYNAMIC NEURONS AND SLOW DYNAMIC INTERACTIONS

As the basic archetypal model we consider a system of Ising spin neurons $\sigma_i \in \{-1, 1\}$, $i \in \{1, \ldots, N\}$, interacting via continuous-valued symmetric interactions, $J_{ij}$, which themselves evolve in response to the states of the neurons. The neurons are taken to have a stochastic field-alignment dynamics which is fast compared with the evolution rate of the interactions $J_{ij}$, such that on the time-scale of $J_{ij}$-dynamics the neurons are effectively in equilibrium according to a Boltzmann distribution,

$$P_{\{J_{ij}\}}(\{\sigma_i\}) \propto \exp\left[-\beta H_{\{J_{ij}\}}(\{\sigma_i\})\right] \qquad (1)$$

where

$$H_{\{J_{ij}\}}\left(\{\sigma_i\}\right) = -\sum_{i<j} J_{ij}\sigma_i\sigma_j \qquad (2)$$

and the subscript $\{J_{ij}\}$ indicates that the $\{J_{ij}\}$ are to be considered as quenched variables. In practice, several specific types of dynamics which obey detailed balance lead to the equilibrium distribution (1), such as a Markov process with single-spin flip Glauber dynamics [1]. The quantity $\beta$ is an inverse temperature characterizing the stochastic gain.

For the $J_{ij}$ dynamics we choose the form

$$\tau \frac{d}{dt} J_{ij} = \frac{1}{N}\langle\sigma_i\sigma_j\rangle_{\{J_{ij}\}} - \mu J_{ij} + \frac{1}{\sqrt{N}}\eta_{ij}(t) \qquad (i<j) \qquad (3)$$

where $\langle\ldots\rangle_{\{J_{ij}\}}$ refers to a thermodynamic average over the distribution (1) with the effectively instantaneous $\{J_{ij}\}$, and $\eta_{ij}(t)$ is a stochastic Gaussian white noise of zero mean and correlation

$$\langle\eta_{ij}(t)\eta_{kl}(t')\rangle = 2\tau\tilde{\beta}^{-1}\delta_{(ij),(kl)}\delta(t-t')$$

The first term on the right-hand side of (3) is inspired by the Hebbian process in neural tissue in which synaptic efficacies are believed to grow locally in response to the simultaneous activity of pre- and post-synaptic neurons [2]. The second term acts to limit the magnitude of $J_{ij}$; $\tilde{\beta}$ is the characteristic inverse temperature of the interaction system. (A related interaction dynamics without the noise term, equivalent to $\tilde{\beta} = \infty$, was introduced by Shinomoto [3]; the anti-Hebbian version of the above coupled dynamics was studied in layered systems by Jonker et al. [4, 5].)

Substituting for $\langle\sigma_i\sigma_j\rangle$ in terms of the distribution (1) enables us to re-write (3) as

$$N\tau\frac{d}{dt}J_{ij} = -\frac{\partial}{\partial J_{ij}}\mathcal{H}\left(\{J_{ij}\}\right) + \sqrt{N}\eta_{ij}(t) \qquad (4)$$

where the effective Hamiltonian $\mathcal{H}\left(\{J_{ij}\}\right)$ is given by

$$\mathcal{H}\left(\{J_{ij}\}\right) = -\frac{1}{\beta}\ln Z_\beta\left(\{J_{ij}\}\right) + \frac{1}{2}\mu N\sum_{i<j} J_{ij}^2 \qquad (5)$$

where $Z_\beta\left(\{J_{ij}\}\right)$ is the partition function associated with (2):

$$Z_\beta\left(\{J_{ij}\}\right) = \operatorname*{Tr}_{\{\sigma_i\}} \exp\left[-\beta H_{\{J_{ij}\}}\left(\{\sigma_i\}\right)\right].$$

## 2    COUPLED SYSTEM IN THERMAL EQUILIBRIUM

We now recognise (4) as having the form of a Langevin equation, so that the equilibrium distribution of the interaction system is given by a Boltzmann form. Henceforth, we concentrate on this equilibrium state which we can characterize by a partition function $\tilde{Z}_{\tilde{\beta}}$ and an associated 'free energy' $\tilde{F}_{\tilde{\beta}}$:

$$\tilde{Z}_{\tilde{\beta}} \equiv \int \prod_{i<j} dJ_{ij} \left[Z_\beta\left(\{J_{ij}\}\right)\right]^n \exp\left[-\frac{1}{2}\tilde{\beta}\mu N\sum_{i<j} J_{ij}^2\right] \qquad \tilde{F}_{\tilde{\beta}} \equiv -\tilde{\beta}^{-1}\ln\tilde{Z}_{\tilde{\beta}} \qquad (6)$$

where $n \equiv \tilde{\beta}/\beta$. We may use $\tilde{Z}_{\tilde{\beta}}$ as a generating functional to produce thermodynamic averages of state variables $\Phi(\{\sigma_i\}; \{J_{ij}\})$ in the combined system by adding suitable infinitesimal source terms to the neuron Hamiltonian (2):

$$H_{\{J_{,j}\}}(\{\sigma_i\}) \rightarrow H_{\{J_{,j}\}}(\{\sigma_i\}) + \lambda\Phi(\{\sigma_i\}; \{J_{ij}\})$$

$$
\begin{aligned}
\lim_{\lambda \to 0} \frac{\partial \tilde{F}_{\tilde{\beta}}}{\partial \lambda} &= \overline{\langle\Phi(\{\sigma_i\}; \{J_{ij}\})\rangle}_{\{J_{,j}\}} \\
&\equiv \frac{\int \prod_{i<j} dJ_{ij} \, \langle\Phi(\{\sigma_i\}; \{J_{ij}\})\rangle_{\{J_{,j}\}} e^{-\tilde{\beta}\mathcal{H}(\{J_{,j}\})}}{\int \prod_{i<j} dJ_{ij} \, e^{-\tilde{\beta}\mathcal{H}(\{J_{,j}\})}}
\end{aligned}
\tag{7}
$$

where the bar refers to an average over the asymptotic $\{J_{ij}\}$ dynamics.

The form (6) with $n \to 0$ is immediately reminiscent of the effective partition function which results from the application of the replica trick to replace $\ln Z$ by $\lim_{n\to 0}\frac{1}{n}(Z^n - 1)$ in dealing with a quenched average for the infinite-ranged spin-glass [6], while $n = 1$ relates to the corresponding annealed average, although we note that in the present model the time-scales for neuron and interaction dynamics remain completely disparate. These observations correlate with the identification of $n$ with $\tilde{\beta}/\beta$, which implies that $n \to 0$ corresponds to a situation in which the interaction dynamics is dominated by the stochastic term $\eta_{ij}(t)$, rather than by the behaviour of the neurons, while for $n = 1$ the two characteristic temperatures are the same. For $n \to \infty$ the influence of the neurons on the interaction dynamics dominates. In fact, any real $n$ is possible by tuning the ratio between the two $\beta$'s. In the formulation presented in this paper $n$ is always non-negative, but negative values are possible if the Hebbian rule of (3) is replaced by an anti-Hebbian form with $\langle\sigma_i\sigma_j\rangle$ replaced by $-\langle\sigma_i\sigma_j\rangle$ (the case of negative $n$ is being studied by Mézard and co-workers [7]).

The model discussed above is range-free/infinite-ranged and can therefore be analyzed in the thermodynamic limit $N \to \infty$ by the replica mean-field theory as devised for the Sherrington-Kirkpatrick spin-glass [6, 8, 9]. This can be developed precisely for integer $n$ [6, 8, 9, 10] and analytically continued. In the usual manner there enters a spin-glass order parameter

$$q^{\gamma\delta} = \overline{\langle\sigma_i^{\gamma}\sigma_i^{\delta}\rangle}_{\{J_{,j}\}} \qquad (\gamma \neq \delta)$$

where the superscripts are replica labels. $q^{\gamma\delta}$ is given by the extremum of

$$F(\{q^{\gamma\delta}\}) = -\frac{\tilde{\beta}}{2\mu n^2}\sum_{\gamma<\delta}[q^{\gamma\delta}]^2 + \ln \operatorname*{Tr}_{\{\sigma^{\gamma}\}} \exp\left[\frac{\tilde{\beta}}{\mu n^2}\sum_{\gamma<\delta}\sigma^{\gamma}q^{\gamma\delta}\sigma^{\delta}\right]$$

while $\tilde{Z}_{\tilde{\beta}}$ is proportional to $\exp\left[N\operatorname{extr}F(\{q^{\gamma\delta}\})\right]$. In the replica-symmetric region (or ansatz) one assumes $q^{\gamma\delta} = q$.

We will first choose as the independent variables $n$ and $\beta$ and briefly discuss the phase picture of our model (full details can be found in [11]). The system exhibits a transition from a paramagnetic state ($q = 0$) to an ordered state ($q > 0$) at a critical $\beta_c(n)$. For $n \leq 2$ this transition is second order at $\beta_c = 1$, down to the SK

spin-glass limit, $n \to 0$, but for $n > 2$ the coupled dynamics leads to a qualitative, as well as quantitative, change to first order. Replica symmetry is stable above a critical value $n_c(\beta)$, at which there is a de Almeida-Thouless (AT) transition (c.f. Kondor [12]). As expected from spin-glass studies, $n_c(\beta)$ goes to zero as $\beta \downarrow 1$ but rises for larger $\beta$, having a maximum of order 0.3 at $\beta$ of order 2. Thus, for $n > n_c(\mathrm{max}) \approx 0.3$ there is no instability against small replica-symmetry breaking fluctuations, while for smaller $n$ there is re-entrance in this stability. The transition from a paramagnetic to an ordered state and the onset of local RS instability for various temperatures is shown in Figure 1.

## 3   EXTERNAL FIELDS

Several simple modifications of the above model are possible. One consists of adding external fields to the spin dynamics and/or to the interaction dynamics, by making the substitutions

$$H_{\{J_{i,j}\}}\left(\{\sigma_i\}\right) \to H_{\{J_{i,j}\}}\left(\{\sigma_i\}\right) - \sum_i \theta_i \sigma_i$$

$$\mathcal{H}\left(\{J_{ij}\}\right) \to \mathcal{H}\left(\{J_{ij}\}\right) - \sum_{i<j} J_{ij} K_{ij}$$

in (2) and (5) respectively. These external fields may be viewed as generating fields in the sense of (7); for example

$$-\frac{\partial \tilde{F}}{\partial \theta_i} = \overline{\langle \sigma_i \rangle} \qquad\qquad -\frac{\partial^2 \tilde{F}}{\partial \theta_i \partial \theta_j} = \tilde{\beta}\left[\overline{\langle \sigma_i \rangle \langle \sigma_j \rangle} - \overline{\langle \sigma_i \rangle}\ \overline{\langle \sigma_j \rangle}\right] + \beta \left[\overline{\langle \sigma_i \sigma_j \rangle} - \overline{\langle \sigma_i \rangle \langle \sigma_j \rangle}\right]$$

$$-\frac{\partial \tilde{F}}{\partial K_{ij}} = \overline{J_{ij}} \qquad\qquad -\frac{\partial^2 \tilde{F}}{\partial K_{ij} \partial K_{kl}} = \tilde{\beta}\left[\overline{J_{ij} J_{kl}} - \overline{J_{ij}}\ \overline{J_{kl}}\right]$$

For neural network models a natural first choice for the external fields would be $\theta_i \equiv h\xi_i$ and $K_{ij} \equiv K\xi_i\xi_j$, $\xi_i \in \{-1, 1\}$, where the $\xi_i$ are quenched random variables corresponding to an imposed pattern. Without loss of generality all the $\xi_i$ can be taken as $+1$, via the gauge transformation $\sigma_i \to \sigma_i\xi_i$, $J_{ij} \to J_{ij}\xi_i\xi_j$. Henceforth we shall make this choice. The neuron perturbation field $h$ induces a finite 'magnetization' characterized by a new order parameter

$$m^\alpha = \overline{\langle \sigma_i^\alpha \rangle}$$

which is independent of $\alpha$ in the replica-symmetric assumption (which turns out to be stable with respect to variation in this parameter). As in the case of the spin-glass, there is now a critical surface in $(h, n, \beta)$ space characterizing the onset of replica symmetry breaking. In introducing the interaction perturbation field $K$ we find that $K/\mu$ is the analogue of the mean exchange $J_0$ in the SK spin-glass model, $\tilde{J}^2 \equiv (\beta n \mu)^{-1}$ being the analogue of the variance. If large enough, this field leads to a spontaneous 'ferromagnetic' order.

Again we find further examples of both second and first order transitions (details can be found in [11]). For the paramagnetic (P; $m = 0$, $q = 0$) to ferromagnetic (F; $m \neq 0$, $q \neq 0$) case, the transition is second order at the SK value $\beta J_0 = 1$ so long as $(\beta \tilde{J})^{-2} \geq 3n - 2$. Only when $(\beta \tilde{J})^{-2} < 3n - 2$ do the interaction dynamics

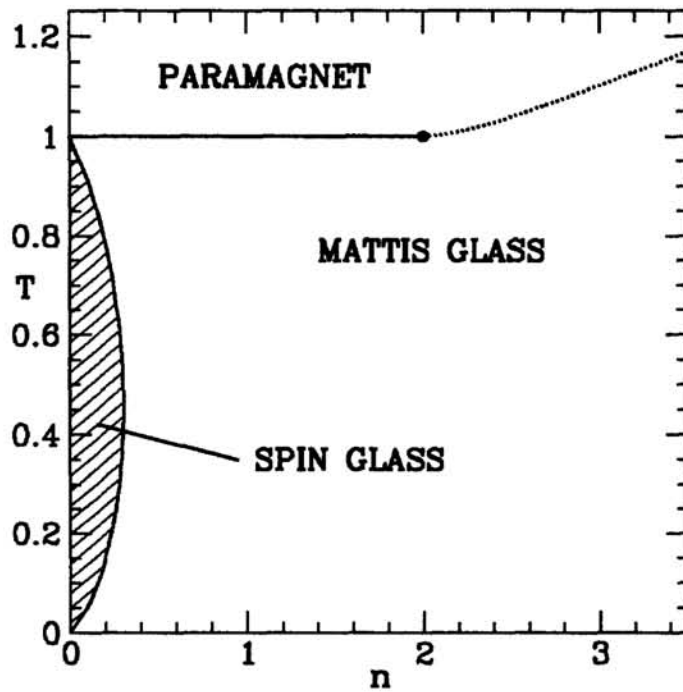

Figure 1: Phasediagram for $\tilde{J} = 1$. Dotted line: first order transition, solid line: second order transition. The separation between Mattis-glass and spin-glass phase is defined by the de Almeida-Thouless instability

influence the transition, changing it to first order at a lower temperature. Regarding the ferromagnetic to spin-glass (SG; $m = 0$, $q \neq 0$) transition, this exhibits both second order (lower $J_0$) and first order (higher $J_0$) sections separated by a tricritical point for $n$ less than a critical value of the order of 3.3. This tricritical point exhibits re-entrance as a function of $n$.

# 4  COMPARISON BETWEEN COUPLED DYNAMICS AND SK MODEL

In order to clarify the differences, we will briefly summarize the two routes that lead to an SK-type replica theory:

Coupled Dynamics:

Fast Ising spin neurons + slow dynamic interactions,

$$\frac{d}{dt} J_{ij} = \frac{1}{N} \langle \sigma_i \sigma_j \rangle_{\{J_{ij}\}} + \frac{K}{N} - \mu J_{ij} + \text{GWN}$$

Free energy:

$$\tilde{f} \equiv -\frac{1}{\tilde{\beta} N} \log \tilde{Z}, \quad \tilde{Z} \equiv \int \prod_{i<j} dJ_{ij} \; e^{-\tilde{\beta}\mathcal{H}(\{J_{ij}\})}$$

Define:

$$\tilde{J}_0 \equiv K/\mu, \quad \tilde{J} \equiv [\mu\tilde{\beta}]^{-\frac{1}{2}}$$

Thermodynamics:

$$N \to \infty: \qquad \tilde{f} = -\frac{1}{\beta n} \text{extr}\, G\left(\{q^{\gamma\delta}\}; \{m^\gamma\}\right) + \text{const.}$$

SK spin-glass:

Ising spins + fixed random interactions,

$$P(J_{ij}) \equiv [2\pi J^2]^{-\frac{1}{2}} e^{-\frac{1}{2}[J_{ij} - J_0]^2 / J^2}$$

Free energy:

$$f \equiv -\frac{1}{\beta N} \log Z = -\frac{1}{\beta N} \lim_{n \to 0} \frac{1}{n} [Z^n - 1]$$

Selt-averaging:

$$f \; \to \; \overline{f} \equiv \langle f \rangle_{\{J_{ij}\}}$$

Physical scaling:

$$J_0 = \tilde{J}_0 / N, \quad J = \tilde{J}/\sqrt{N}$$

Thermodynamics:

$$N \to \infty: \qquad \overline{f} = -\lim_{n \to 0} \frac{1}{\beta n} \text{extr}\, G\left(\{q^{\gamma\delta}\}; \{m^\gamma\}\right) + \text{const.}$$

# 5   DISCUSSION

We have obtained a solvable model with which a coupled dynamics of fast stochastic neurons and slow dynamic interactions can be studied analytically. Furthermore it presents the replica method from a novel perspective, provides a direct interpretation of the replica dimension $n$ in terms of parameters controlling dynamical processes and leads to new phase transition characters. As a model for neural learning the specific example analyzed here is however only a first step, with $h$ and $K$ as introduced corresponding to only a single pattern. Its adaptation to treat many patterns is the next challenge.

One type of generalization is to consider the whole system as of lower connectivity with only pairs of connected sites being available for interaction upgrade. For example, the system could be on a lattice, in which case the corresponding coupled partition function will have the usual greater complication of a finite-dimensional system, or randomly connected with each bond present with a probability $C/N$, in which case there results an analogue of the Viana-Bray [13] spin-glass. In each of these cases the explicit factors involving $N$ in the $\{J_{ij}\}$ dynamics (3) should be removed (their presence or absence being determined by the need for statistical relevance and physical scaling).

Yet another generalization is to higher order interactions, for example to $p$-neuron ones:

$$H_{\{J\}}\left(\{\sigma_i\}\right) = -\sum_{i_1,\dots,i_p} J_{i_1,\dots,i_p}\sigma_{i_1}\sigma_{i_2}\dots\sigma_{i_p}$$

with corresponding interaction dynamics

$$\tau\frac{d}{dt}J_{i_1,\dots i_p} = \frac{1}{N}\langle\sigma_{i_1}\dots\sigma_{i_p}\rangle_{\{J\}} - \mu J_{i_1,\dots,i_p} + \frac{1}{\sqrt{N}}\eta_{i_1,\dots,i_p}(t)$$

or to more complex neuron types.

If the symmetry-breaking fields $K_{ij}$ in the interaction dynamics are choosen at random, we obtain a curious theory in which we find replicas on top of replicas (the replica trick would be used to deal with the quenched disorder of the $K_{ij}$, for a model in which replicas are already present due to the coupled dynamics).

Finally, our approach can in fact be generalized to *any* statistical mechanical system which in equilibrium is described by a Boltzmann distribution in which the Hamiltonian has (adiabatically slowly) evolving parameters. By choosing these parameters to evolve according to an appropriate Langevin process (involving the free energy of the underlying fast system) one always arrives at a replica theory describing the coupled system in equilibrium.

**Acknowledgements**

Financial support from the U.K. Science and Engineering Research Council under grants 9130068X and GR/H26703, from the European Community under grant S/SC1*915121, and from Jesus College, Oxford, is gratefully acknowledged.

# References

[1] Glauber R.J. (1963) *J. Math. Phys.* **4** 294

[2] Hebb D.O. (1949) *'The Organization of Behaviour'* (Wiley, New York)

[3] Shinomoto S. (1987) *J. Phys. A: Math. Gen.* **20** L1305

[4] Jonker H.J.J. and Coolen A.C.C. (1991) *J. Phys. A: Math. Gen.* **24** 4219

[5] Jonker H.J.J., Coolen A.C.C. and Denier van der Gon J.J. (1993) *J. Phys. A: Math. Gen.* **26** 2549

[6] Sherrington D. and Kirkpatrick S. (1975) *Phys. Rev. Lett.* **35** 1792

[7] Mézard M. *private communication*

[8] Kirkpatrick S. and Sherrington D. (1978) *Phys. Rev. B* **17** 4384

[9] Mézard M., Parisi G. and Virasoro M.A. (1987) *'Spin Glass Theory and Beyond'* (World Scientific, Singapore)

[10] Sherrington D. (1980) *J. Phys. A: Math. Gen.* **13** 637

[11] Penney R.W., Coolen A.C.C. and Sherrington D. (1993) *J. Phys. A: Math. Gen.* **26** 3681-3695

[12] Kondor I. (1983) *J. Phys. A: Math. Gen.* **16** L127

[13] Viana L. and Bray A.J. (1983) *J. Phys. C* **16** 6817